# Finding Latent Causes in Causal Networks: an Efficient Approach Based on Markov Blankets

**Jean-Philippe Pellet**[1,2]
jep@zurich.ibm.com
[1] Pattern Recognition and Machine Learning Group
Swiss Federal Institute of Technology Zurich
8092 Zurich, Switzerland

**André Elisseeff**[2]
ael@zurich.ibm.com
[2] Data Analytics Group
IBM Research GmbH
8803 Rüschlikon, Switzerland

## Abstract

Causal structure-discovery techniques usually assume that all causes of more than one variable are observed. This is the so-called causal sufficiency assumption. In practice, it is untestable, and often violated. In this paper, we present an efficient causal structure-learning algorithm, suited for causally insufficient data. Similar to algorithms such as IC* and FCI, the proposed approach drops the causal sufficiency assumption and learns a structure that indicates (potential) latent causes for pairs of observed variables. Assuming a constant local density of the data-generating graph, our algorithm makes a quadratic number of conditional-independence tests w.r.t. the number of variables. We show with experiments that our algorithm is comparable to the state-of-the-art FCI algorithm in accuracy, while being several orders of magnitude faster on large problems. We conclude that MBCS* makes a new range of causally insufficient problems computationally tractable.

**Keywords:** Graphical Models, Structure Learning, Causal Inference.

## 1 Introduction: Task Definition & Related Work

The statistical definition of causality pioneered by Pearl (2000) and Spirtes et al. (2001) has shed new light on how to detect causation. Central in this approach is the automated detection of cause–effect relationships using observational (i.e., non-experimental) data. This can be a necessary task, as in many situations, performing randomized controlled experiments to unveil causation can be impossible, unethical, or too costly. When the analysis deals with variables that cannot be manipulated, being able to learn from data collected by observing the running system is the only possibility.

It turns out that learning the full causal structure of a set of variables is, in its most general form, impossible. If we suppose that the "causal ground truth" can be represented by a directed acyclic graph (DAG) over the variables to analyze, denoted by $\mathbf{V}$, where the arcs denote direct causation, current causal structure-learning algorithms can only learn an equivalence class representing statistically indistinguishable DAGs. This class can be represented by a partially directed acyclic graph (PDAG), where arcs between variables may be undirected, indicating that both directions are equally possible given the data. This is know as the problem of *causal underdetermination* (Pearl, 2000).

Common to most structure-learning algorithms are three important assumptions which ensure the correctness of the causal claims entailed by the returned PDAG (see Scheines, 1997, for a more extensive discussion of these assumptions and of their implications). First, the *causal Markov condition* states that every variable is independent of its non-effects given its direct causes. It implies that every dependency can be explained by some form of causation (direct, indirect, common cause, or any combination). Second, the *faithfulness condition* demands that the dependencies be DAG-isomorphic; i.e., that there be a DAG whose entailed variable dependencies coincide exactly with

the dependencies found in the data. Third, *causal sufficiency* of the data states that every common cause for two variables in **V** is also in **V**. Causal sufficiency often appears as the most controversial assumption as it is generally considered impossible to ensure that all possible causes are measured—there is no such thing as a closed world. In this paper, we are interested in relaxing causal sufficiency: we do not require the data to contain all common causes of pairs of variables.

Some of the few algorithms that relax causal sufficiency are Inductive Causation* (IC*) by Pearl and Verma (1991); Pearl (2000), and Fast Causal Inference (FCI) by Spirtes et al. (1995, 2001). The kind of graph IC* and FCI return is known as a partial ancestral graph (PAG), which indicate for each link whether it (potentially) is the manifestation of a hidden common cause for the two linked variables. Assuming continuous variables with linear causal influences, Silva et al. (2006) recover hidden variables that are the cause for more than two observed variables, to infer the relationships between the hidden variables themselves. They check additional constraints on the covariance matrix, known as tetrad constraints (Scheines et al., 1995), entailed by special kinds of hidden structures.

There are more specialized techniques to deal with hidden variables. Elidan et al. (2001) look for structural signatures of hidden variables in a learned DAG model. Boyen et al. (1999) describe a technique that looks for violation of the Markov condition to infer the presence of latent variables in Bayesian networks. Once a hidden variable is identified, Elidan and Friedman (2001) discuss how to assign it a given dimensionality to best model its interactions with the observed variables.

In this paper, we describe recent advances in making the PAG-learning task tractable for a wider range of problems, and present the Markov blanket/collidet set (MBCS*) algorithm. In Section 2, we formally describe the PAG-learning task and motivate it with an example. Section 3 describes FCI. We then present MBCS* in Section 4 and compare it experimentally to FCI in Section 5. We finally conclude in Section 6. Correctness proofs are provided in the supplemental material[1].

**Notation**    Throughout this paper, uppercase capitals such as $X$ and $Y$ denote variables or nodes in a graph and sets of variables are set in boldface, such as **V**. $H$ and $L$ (possibly with indices) denote latent (unobserved) variables. Bold lowercase greek characters such as $\pi$ are paths (ordered list of nodes), while the calligraphic letter $\mathcal{G}$ refers to a graph. Finally, we denote conditional independence of $X$ and $Y$ given **Z** by the notation $(X \perp\!\!\!\perp Y \mid \mathbf{Z})$.

## 2    Mixed Ancestral Graphs & Partial Ancestral Graphs

In this section, we first introduce the notation of mixed ancestral graphs (MAGs) and partial ancestral graphs (PAGs) used by Spirtes et al. (1996) and describe how to learn them on a high level. We first review the definition of a V-structure.

**Definition 2.1 (V-structure)** *In a causal DAG, a V-structure is a triplet* $X \rightarrow Z \leftarrow Y$*, where $X$ and $Y$ are nonadjacent. $Z$ is then called an* unshielded collider *for $X$ and $Y$. Its presence implies:*

$$\exists \mathbf{S}_{XY} \subseteq \mathbf{V} \setminus \{X, Y, Z\} : \big( (X \perp\!\!\!\perp Y \mid \mathbf{S}_{XY}) \text{ and } (X \not\perp\!\!\!\perp Y \mid \mathbf{S}_{XY} \cup \{Z\}) \big). \qquad (1)$$

In a V-structure, two causes $X$ and $Y$, which are made independent by $\mathbf{S}_{XY}$, become dependent when conditioned on a common effect $Z$ (or one of its descendants). This is the base fact that allows initial edge orientation in causal structure learning.

Let us now suppose we are learning from data whose (unknown) actual causal DAG is:

$$X \rightarrow Y \leftarrow H_1 \rightarrow H_2 \rightarrow Z \leftarrow W. \qquad (2)$$

Further assume that $H_1$ and $H_2$ are hidden. Assuming the adjacencies $X - Y - Z - W$ have been found, conditional-independence tests will reveal that $(X \perp\!\!\!\perp Z)$ and $(X \not\perp\!\!\!\perp Z \mid Y)$, which is a sufficient condition for the V-structure $X \rightarrow Y \leftarrow Z$. Similarly, $(Y \perp\!\!\!\perp W)$ and $(Y \not\perp\!\!\!\perp W \mid Z)$ is a sufficient condition for the V-structure $Y \rightarrow Z \leftarrow W$. In a DAG like in a PDAG, however, those two overlapping V-structures are incompatible. The simplest DAG compatible with those findings needs the addition of an extra variable $H$:

$$X \rightarrow Y \leftarrow H \rightarrow Z \leftarrow W. \qquad (3)$$

Actually, (3) is the *projection* of the latent structure in (2). In the projection of a latent structure as defined by Pearl (2000), all hidden variables are parentless and have only two direct effects. Verma (1993) proved that any hidden structure has at least one projection. Notice that we cannot recover the information about the two separate hidden variables $H_1$ and $H_2$. In the projection, information can thus be lost with respect to the true latent structure.

Whereas causally sufficient datasets are represented as DAGs and learned as PDAGs to represent independence-equivalent DAGs, the projection of latent structures is represented by special graphs known as mixed ancestral graphs (MAGs) (Spirtes et al., 2001), which allow for bidirected arrows to represent a hidden cause for a pair of variables. Independence-equivalent MAGs are represented by partial ancestral graphs (PAGs). PAGs are thus to MAGs what PDAGs are to DAGs, and structure-learning algorithms like FCI return a PAG.

PAGs allow four kinds of arrows: $\rightarrow$, $\circ\!\!\rightarrow$, $\circ\!\!-\!\!\circ$, and $\leftrightarrow$. $X \rightarrow Y$ in the PAG denotes true causation $X \rightarrow Y$ in the projection; $X \leftrightarrow Y$ indicates the presence of a latent cause $X \leftarrow H \rightarrow Y$ (without excluding direct causation); $X \circ\!\!\rightarrow Y$ denotes either true causation $X \rightarrow Y$ or a latent cause $X \leftarrow H \rightarrow Y$ (or a combination of both); finally, $X \circ\!\!-\!\!\circ Y$ denotes potential causation from $X \rightarrow Y$ or $Y \rightarrow X$ and/or a latent common cause $X \leftarrow H \rightarrow Y$ in the projection, and is thus the most "agnostic link." An asterix as an arrowhead is a wildcard for any of the three possible endpoints of a link, such that $X *\!\!\rightarrow Y$, for instance, means any of $X \rightarrow Y$, $X \circ\!\!\rightarrow Y$, and $X \leftrightarrow Y$. Additionally, we also use the notation $X *\!\!-\!\!* Z *\!\!-\!\!* Y$ to indicate that $Z$ is a *definite noncollider* for $X$ and $Y$, such that any of $X *\!\!\rightarrow Z \rightarrow Y$, $X \leftarrow Z *\!\!\rightarrow Y$, or $X \leftarrow Z \rightarrow Y$ can occur, but not $X *\!\!\rightarrow Z \leftarrow\!\!* Y$.

To illustrate how MAGs and PAGs are related to a latent structure, consider the causal graph shown in Figure 1 (*i*). There, the hidden variable $H$ is a cause for 3 observed variables, and $L$ is a hidden variable in the causal chain from $Z$ to $W$. All other variables are observed. In (*ii*), we show the projection of (*i*): note that we lose information about $L$ and about the fact that $H_1$, $H_2$, and $H_3$ are actually the same variable. The corresponding MAG is shown in (*iii*), and in (*iv*) the PAG that represents the class of independence-equivalent MAGs of which (*iii*) is a member. Note how the causal-underdetermination problem influences PAG learning: for instance, the model shown in (*vi*), if learned as a PAG, will also be represented as in (*iv*). (*v*) is commented on later in the text.

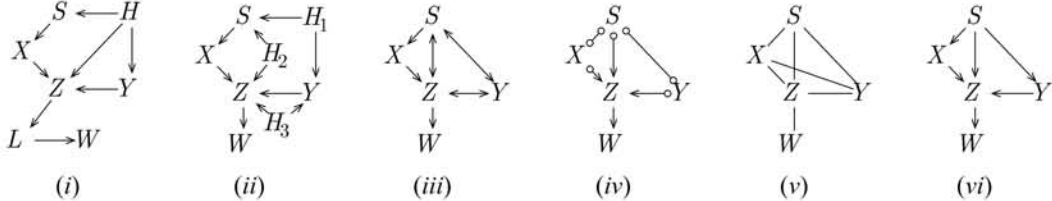

Figure 1: (*i*) Example of a causal structure with the hidden variables $H$ and $L$. (*ii*) Projection of (*i*). (*iii*) MAG representing the projection (*ii*). (*iv*) PAG representing the class of projections that are independence-equivalent to (*iii*). (*v*) The moral graph of (*iii*). (*vi*) Another structure with no hidden variable whose learned PAG is (*iv*).

## 3 Learning PAGs with the FCI Algorithm

This section now turns to the task of learning PAGs with conditional-independence tests and describes shortly the reference algorithm, FCI.

In principle, learning the structure of a PAG is not much different from learning the structure of a PDAG. The main difference is that instead of creating V-structures in a PDAG, we now just add *arrow heads* into the identified colliders, independently of what the other arrow endpoints are. A PAG-learning algorithm could thus operate this way:

1. Adjacencies: insert the "agnostic link" $X \circ\!\!-\!\!\circ Y$ if $\forall \mathbf{S} \subseteq \mathbf{V} \setminus \{X, Y\} : (X \not\perp\!\!\!\perp Y \mid \mathbf{S})$;

2. V-structures: when the condition (1) holds for triplet $(X, Z, Y)$, add arrow heads into $Z$;

3. Orientations: use rules to further orient "agnostic" endpoints wherever possible.

The second difference w.r.t. PDAG learning is in the set of rules applied in Step 3 to further orient the graph. Those rules are detailed in the next subsection.

To the best of our knowledge, the FCI algorithm is regarded as the state-of-the-art implementation of a PAG-learning algorithm. We list its pseudocode in Algorithm 1. The notation $\mathbf{Nb}(X)$ stands for the set of direct neighbors of $X$ in the graph being constructed $\mathcal{G}$ (and potentially changes at each iteration). The set $\mathbf{ExtDSep}(X, Y)$ is the union of $\mathbf{Possible\text{-}D\text{-}Sep}(X, Y)$ and $\mathbf{Possible\text{-}D\text{-}Sep}(Y, X)$. $\mathbf{Possible\text{-}D\text{-}Sep}(X, Y)$ is the set of nodes $Z$ where there is an undirected path $\pi$ between $X$ and $Z$ such that for each subpath $S \leftrightarrow W \leftrightarrow T$ of $\pi$, either (a) $W$ is a collider; or (b) $W$ is not marked as a noncollider and $S, W, T$ are a triangle. (A triangle is a set of three nodes all adjacent to one another.)

We list the orientation rules as a separate procedure in Algorithm 2, as we reuse them in our algorithm. Rule 1 preserves acyclicity. Rule 2 honors the noncollider constraint when one of the two endpoints is an arrowhead. Rule 3 orients double-triangle structures; for instance; it orients $S \circ\!\!\!\rightarrow Z$ in Figure 1 (iii). Rule 4 needs the following definition (Spirtes et al., 1995).

**Definition 3.1 (DDP)** *In a PAG $\mathcal{G}$, $\pi$ is a definite discriminating path (DDP) between $S$ and $Y$ ($S, Y$ nonadjacent) for $Z$ ($Z \neq S, Y$) if and only if $\pi$ is an undirected path between $S$ and $Y$ containing $Z$, $Z$ precedes $Y$ on $\pi$, every vertex $V$ between $S$ and $Z$ on $\pi$ is a collider or a definite noncollider on $\pi$, and:*

*(i) if $V$ and $V'$ are adjacent on $\pi$ and $V'$ is between $V$ and $Z$, then $V \leftrightarrow V'$ on $\pi$;*

*(ii) if $V$ is between $S$ and $Z$ on $\pi$ and $V$ is a collider on $\pi$, then $V \rightarrow Y$ in $\mathcal{G}$, else $Y \leftrightarrow V$ in $\mathcal{G}$.*

Figure 2 shows an example for a DDP and for Rule 4, which produces the orientation $Z \leftarrow\!\circ Y$. For a more extensive justification and a proof of those rules, see Spirtes et al. (1995, 2001).

The time complexity of FCI makes it non-scalable for larger networks. In particular, the two subset searches at lines 5 and 19 of Algorithm 1 are computationally costly in dense networks. In the next section, we present an algorithm that takes another approach at PAG learning to tackle problems larger than those that FCI can handle.

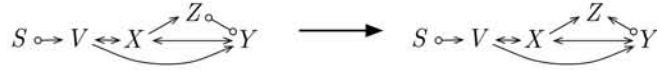

Figure 2: Path $\pi = \langle S, V, X, Z, Y \rangle$ is a DDP for $Z$. Rule 4 adds an arrow head into $Z$ if $Z \notin \mathbf{S}_{SY}$.

## 4 Efficient Structure Learning with the MBCS* Algorithm

In this section, we propose a PAG-learning algorithm, MBCS*, which is more efficient than FCI in the sense that it performs much fewer conditional-independence tests, whose average conditioning-set size is smaller. We show in Section 5 that MBCS* compares very favorably to FCI on test networks in terms of computational tractability, while reaching similar accuracy. Pseudocode for MCBS* is listed in Algorithm 3.

MBCS* proceeds in three steps: first, it detects the Markov blankets for each variable; second, it examines the triangle structures to identify colliders and noncolliders; finally, it uses the same orientation rules as FCI to obtain the maximally oriented PAG. We detail the first two steps below; the orientation rules are the same as for FCI.

### 4.1 Step 1: Learning the Markov Blanket

The first phase of MBCS* builds an undirected graph where each variable is connected to all members of its Markov blanket.

**Definition 4.1 (Markov blanket)** *The Markov blanket of a node $X$ is the smallest set of variables $\mathbf{Mb}(X)$ such that $\forall Y \in \mathbf{V} \setminus \mathbf{Mb}(X) \setminus \{X\} : (X \perp\!\!\!\perp Y \mid \mathbf{Mb}(X))$.*

Assuming faithfulness, $\mathbf{Mb}(X)$ is unique. In a DAG, it corresponds to the parents, children, and children's parents (spouses) of $X$. We extend this to MAGs.

---

**Algorithm 1**  $\mathcal{G} = \text{FCI}(\mathbf{V}, I)$

---

| **Input:** | $\mathbf{V}$ : | set of observed variables |
| | $I$ : | a conditional-independence oracle, called with the notation $(\cdot \perp\!\!\!\perp \cdot \mid \cdot)$ |
| **Output:** | $\mathcal{G}$ : | maximally oriented partial ancestral graph |

1: $\mathcal{G} \leftarrow$ fully connected graph over $\mathbf{V}$
2: $i \leftarrow 0$

    *// Detect adjacencies*
3: **while** $\exists(X - Y)$ s.t. $|\text{Nb}(X)| > i$ **do**
4:    **for each** $X - Y$ s.t. $|\text{Nb}(X)| > i$ **do**
5:        **for each** $\mathbf{S} \subseteq \text{Nb}(X) \setminus \{Y\}$ of size $i$ **do**
6:            **if** $(X \perp\!\!\!\perp Y \mid \mathbf{S})$ **then**
7:                remove link $X - Y$ from $\mathcal{G}$
8:                $\mathbf{S}_{XY}, \mathbf{S}_{YX} \leftarrow \mathbf{S}$
9:                **break** from loop line 4
10:            **end if**
11:        **end for**
12:    **end for**
13:    $i \leftarrow i + 1$
14: **end while**
15: **for each** $X - Z - Y$ s.t. $X, Y$ nonadjacent **do**
16:    **if** $Z \notin \mathbf{S}_{XY}$ **then** orient as $X \to Z \leftarrow Y$
17: **end for**

    *// Detect additional adjacencies*
18: **for each** pair of adjacent variables $X, Y$ **do**
19:    **for each** $\mathbf{S} \subseteq \text{ExtDSep}(X,Y) \setminus \{X,Y\}$ **do**
20:        **if** $(X \perp\!\!\!\perp Y \mid \mathbf{S})$ **then**
21:            remove link $X - Y$ from $\mathcal{G}$
22:            $\mathbf{S}_{XY}, \mathbf{S}_{YX} \leftarrow \mathbf{S}$
23:            **break** from loop line 18
24:        **end if**
25:    **end for**
26: **end for**
27: orient every link as $\circ\!\!-\!\!\circ$

    *// Orient V-structures*
28: **for each** $X \leftrightarrow Z \leftrightarrow Y$ s.t. $X, Y$ nonadjacent **do**
29:    **if** $Z \notin \mathbf{S}_{XY}$ **then** orient as $X \rightarrowtail Z \leftarrowtail Y$
30:    **else** mark $Z$ as noncollider: $X \leftrightarrow \underline{Z} \leftrightarrow Y$
31: **end for**

32: **return** $\text{ORIENTMAXIMALLY}(\mathcal{G}, \forall(X,Y) : \mathbf{S}_{XY})$

---

**Algorithm 2**  $\mathcal{G} = \text{ORIENTMAXIMALLY}(\mathcal{G},$ a list of sets $\mathbf{S}_{XY})$

---

| **Input:** | $\mathcal{G}$ : | partial ancestral graph |
| | $\mathbf{S}_{XY}$ : | for (some) nonadjacent pairs $(X, Y)$: a $d$-separating set of variables |
| **Output:** | $\mathcal{G}$ : | maximally oriented partial ancestral graph |

1: **while** $\mathcal{G}$ is changed by some rule **do**
2:    **for each** $X \leftarrow\!\!\circ Y$ such that there is a directed path from $X$ to $Y$ **do** orient as $X \rightarrowtail Y$      *// Rule 1*
3:    **for each** $X \rightarrowtail \underline{Z} \circ\!\!\rightarrow Y$ **do** orient as $X \rightarrowtail Z \to Y$      *// Rule 2*
4:    **for each** $X \rightarrowtail \overline{Z} \leftarrowtail Y$ with $S \leftarrow\!\!\circ Z$ and $S \in \mathbf{S}_{XY}$ **do** orient as $S \rightarrowtail Z$      *// Rule 3*
5:    **for each** definite discriminating path $\pi$ between $S$ and $Y$ for $Z$ **do**      *// Rule 4*
6:        **if** $X \leftrightarrow Y$ where $X$ is adjacent to $Z$ on $\pi$ and $X, Z, Y$ are a triangle **then**
7:            **if** $\mathbf{S}_{SY}$ exists and $Z \notin \mathbf{S}_{SY}$ **then** orient as $X \rightarrowtail Z \leftarrowtail Y$
8:            **else** mark $Z$ as a noncollider $X \leftrightarrow \underline{Z} \leftrightarrow Y$
9:        **end if**
10:    **end for**
11: **end while**

---

**Algorithm 3**  $\mathcal{G} = \text{MBCS*}(\mathbf{V}, I)$

---

| **Input:** | $\mathbf{V}$ : | set of observed variables |
| | $I$ : | a conditional-independence oracle, called with the notation $(\cdot \perp\!\!\!\perp \cdot \mid \cdot)$ |
| **Output:** | $\mathcal{G}$ : | maximally oriented partial ancestral graph |

    *// Initialization*
1: $\mathcal{G} \leftarrow$ empty graph over $\mathbf{V}$

    *// Find Markov blankets (Grow-Shrink)*
2: **for each** $X \in \mathbf{V}$ **do**
3:    $\mathbf{S} \leftarrow$ empty set of Markov blanket variables
4:    **while** $\exists Y \in \mathbf{V} \setminus \{X\}$ s.t. $(X \not\perp\!\!\!\perp Y \mid \mathbf{S})$ **do**
5:        add $Y$ to $\mathbf{S}$
6:    **while** $\exists Y \in \mathbf{S}$ s.t. $(X \perp\!\!\!\perp Y \mid \mathbf{S} \setminus \{Y\})$ **do**
7:        remove $Y$ from $\mathbf{S}$
8:    **for each** $Y \in \mathbf{S}$ **do** add link $X \circ\!\!-\!\!\circ Y$
9: **end for**

    *// Add noncollider constraints*
10: **for each** $X \circ\!\!-\!\!\circ Z \circ\!\!-\!\!\circ Y$ s.t. $X, Y$ nonadjacent **do**
11:    mark as noncollider $X \circ\!\!-\!\!\circ \underline{Z} \circ\!\!-\!\!\circ Y$
12: **end for**

    *// Adjust local structures (Collider Set search)*
13: $\mathbf{C} \leftarrow$ empty list of collider-orientation directives
14: **for each** $X \leftrightarrow Y$ in a fully connected triangle **do**
15:    **if** $\exists$collider set $\mathbf{Z} \subseteq \text{Tri}(X - Y)$ **then**
16:        $\mathbf{S}_{XY} \leftarrow d$-separating set for $(X, Y)$
17:        remove link $X \leftrightarrow Y$ from $\mathcal{G}$
18:        **for each** $Z \in \mathbf{Z}$ **do**
19:            add ordered triplet $(X, Z, Y)$ to $\mathbf{C}$
20:        **for each** $Z \in \mathbf{S}_{XY}$ **do**
21:            mark as noncollider $X \leftrightarrow \underline{Z} \leftrightarrow Y$
22:    **end if**
23: **end for**
24: **for each** orientation directive $(X, Z, Y) \in \mathbf{C}$ **do**
25:    **if** $X \leftrightarrow Z \leftrightarrow Y$ **then** orient as $X \rightarrowtail Z \leftarrowtail Y$
26: **end for**
27: **return** $\text{ORIENTMAXIMALLY}(\mathcal{G}, \forall(X,Y) : \mathbf{S}_{XY})$

**Property 4.2** *In a faithful MAG, the Markov blanket* $\mathbf{Mb}(X)$ *of a node* $X$ *is the set of parents, children, children's parents (spouses) of* $X$, *as well as the district of* $X$ *and of the children of* $X$, *and the parents of each node of these districts, where the district of a node* $Y$ *is the set of all nodes reachable from* $Y$ *using only bidirected edges. (Proof in supplemental material.)*

We use algorithmic ideas from Margaritis and Thrun (1999) to learn the Markov blanket of a node with a linear number of conditional-independence tests (proof in the supplemental material of Margaritis and Thrun, 1999). This technique is used in lines 3 to 6 of Algorithm 3. The resulting graph is an undirected graph called *moral graph* where each node is connected to its Markov blanket. Therefore, it contains spurious links to its spouses, to members of its district, to members of its children's district, and to parents of nodes in those districts, which we all call *SD links* (for Spouse/District). Removal of those links is done in the second step of MBCS*.

## 4.2 Step 2: Removing the SD Links

In the second step of MBCS*, each undirected edge must be identified as either an SD link to be removed, or a true link of the original MAG to be kept. Direct parents and children are dependent given any conditioning set, while spouses and district members (and their parents) can be made independent. For each link $X - Y$, a search is thus performed to try to $d$-separate the two connected nodes. This search can be limited to the smallest of the Markov blankets of $X$ and $Y$, as by definition they contain all nodes that can minimally make them independent from each other, provided they are linked by an SD link. If such a $d$-separating set $\mathbf{S}_{XY}$ is found, the link is removed. Interestingly, identifying a $d$-separating set $\mathbf{S}_{XY}$ also identifies the collider set for $X$ and $Y$.

**Definition 4.3 (Collider set)** *In an undirected graph* $\mathcal{G}$ *over* $\mathbf{V}$, *let* $\mathbf{Tri}(X - Y)$ $(X, Y$ *adjacent) be the set of all vertices that form a triangle with* $X$ *and* $Y$. *Suppose that* $\mathcal{G}$ *is the moral graph of the DAG or MAG representing the causal structure of a faithful dataset. A set of vertices* $\mathbf{Z} \subseteq \mathbf{Tri}(X - Y)$ *then has the* Collider set *property for the pair* $(X, Y)$ *if it is the largest set that fulfills*

$$\exists \mathbf{S}_{XY} \subseteq \mathbf{V} \setminus \{X, Y\} \setminus \mathbf{Z} : (X \perp\!\!\!\perp Y \mid \mathbf{S}_{XY})$$
$$and \ \ \forall Z \in \mathbf{Z} : (X \not\!\perp\!\!\!\perp Y \mid \mathbf{S}_{XY} \cup \{Z\}).$$

Collider sets are useful because each node in them satisfies the property of a collider (1) and reveals a V-structure. Suppose $(X \perp\!\!\!\perp Y \mid \mathbf{S}_{XY})$: then each node $Z$ (connected by a non-SD link to both $X$ and $Y$) not in $\mathbf{S}_{XY}$ is a collider. This follows from the fact that for each path $X \leftrightarrow Z \leftrightarrow Y$ where $Z \notin \mathbf{S}_{XY}$, the only structural possibility is to have arrow head pointing into $Z$ by the definition of $d$-separation (Pearl, 1988). Similarly, if $Z \in \mathbf{S}_{XY}$, then any orientation is possible save for a collider. Those two types of constraints appear in lines 19 and 21 of Algorithm 3, respectively. Note that more noncollider constraints are added in line 11: in the case $X \circ\!\!-\!\!\circ Z \circ\!\!-\!\!\circ Y$ with $X, Y$ nonadjacent, we know that $Z$ cannot be a V-structure owing to the following lemma.

**Lemma 4.4** *In the moral graph* $\mathcal{G}_m$ *of a DAG or a MAG* $\mathcal{G}$, *whenever the pattern* $X \leftrightarrow Z \leftrightarrow Y$ *occurs in* $\mathcal{G}$, *then* $X$ *and* $Y$ *are linked in* $\mathcal{G}_m$. *(Proof in supplemental material.)*

In practice, the search for collider sets and simultaneously for $d$-separating sets in lines 15 and 16 is performed following the implementation proposed by Pellet and Elisseeff (2008). They also discuss why V-structure orientations must be delayed to line 25 instead of being made immediately in line 19.

In the supplemental material to this paper, we prove that MBCS* correctly identifies all adjacencies and V-structures. The final orientation step (Algorithm 2) requires $d$-separating-set information for Rules 3 and 4: we also prove that MBCS* provides all necessary information.

# 5 Experimental Evaluation

We now compare FCI and MBCS* with a series of experiments. We took two standard benchmark networks, ALARM and HAILFINDER, and for each of them, chose to hide 0, 1, 2, and 3 variables, creating in total 8 learning problems. On a first series on experiments, the algorithms were run with a $d$-separation oracle, which is equivalent to perfect conditional-independence tests. Conditioning

Table 1: Comparison of MBCS* and FCI where conditional-independence tests are done using a $d$-separation oracle. We report the number of tests $t$; the weighted number of tests $wt$, where each test contributes to $wt$ a summand equal to the size of its conditioning set; and the ratio of $t$ for FCI over the $t$ for MBCS*: $r \approx t\,(\text{FCI})\,/t\,(\text{MBCS*})$.

| Alg. | | ALARM | | | | HAILFINDER | | |
|---|---|---|---|---|---|---|---|---|
| | #hid. v. | $t$ | $wt$ | $r$ | #hid. v. | $t$ | $wt$ | $r$ |
| MBCS* | 0 | 2,237 | 12,123 | $\frac{1}{4}$ | 0 | 5,333 | 35,841 | $\frac{1}{423}$ |
| FCI | | 9,340 | 27,666 | | | 2,254,774 | 20,153,894 | |
| MBCS* | 1 | 3,397 | 18,113 | $\frac{1}{6}$ | 1 | 6,516 | 42,379 | $\frac{1}{353}$ |
| FCI | | 21,497 | 95,497 | | | 2,302,707 | 20,448,775 | |
| MBCS* | 2 | 5,208 | 27,576 | $\frac{1}{6}$ | 2 | 7,205 | 46,291 | $\frac{1}{322}$ |
| FCI | | 31,018 | 145,322 | | | 2,324,503 | 20,608,841 | |
| MBCS* | 3 | 7,527 | 42,133 | $\frac{1}{30}$ | 3 | 18,244 | 117,209 | $\frac{1}{143}$ |
| FCI | | 231,096 | 1,612,106 | | | 2,622,312 | 22,888,622 | |

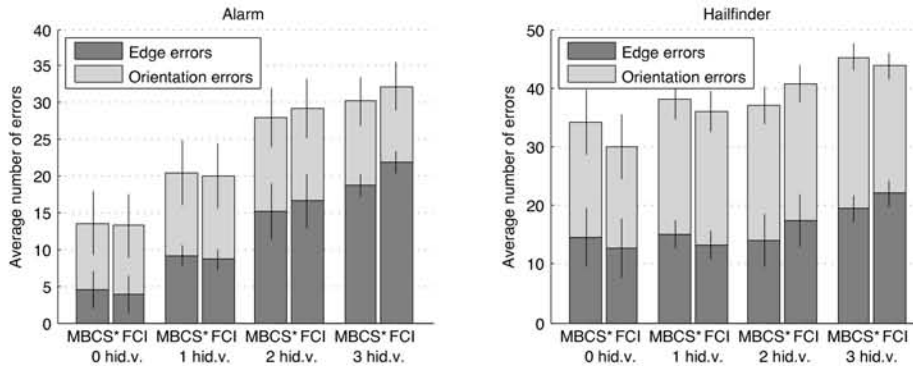

Figure 3: Comparison of MBCS* and FCI where conditional-independence tests are done using Fisher's $z$-test. We compare the number of edge errors (missing/extraneous) and orientation errors (including missing/extraneous hidden variables). Error bars show the standard deviation over the 5 runs.

on hidden variables was prohibited. The results are listed in Table 1. In a second series, multivariate Gaussian datasets (with 500 datapoints) were sampled from the networks and data corresponding to the hidden variables were removed. The algorithms were run with Fisher's $z$-test on partial correlation as conditional-independence test. This was repeated 5 times for each learning problem.[2] For FCI, we used the authors' implementation in TETRAD (Scheines et al., 1995). MBCS* was implemented in Matlab. See Figure 3 for the comparison.

Table 1 shows in the columns named $t$ that MBCS* makes up to 3 orders of magnitude fewer conditional-independence tests than FCI on the tested networks. As the number of tests alone does not reflect the quality of the algorithm, we also list in the $wt$ column a weighted sum of tests, where each test is weighted by the size of its conditioning set. As the ALARM network becomes denser by hiding certain variables, the difference between FCI and MBCS* becomes even more apparent. The inverse phenomenon is to be observed for HAILFINDER, where the difference between FCI and MBCS* gets smaller: this is because this network is more densely connected, and both algorithms exhibit a behavior gradually evolving towards the worst case of the fully connected graph. FCI slowly "catches up" with MBCS* in those circumstances.

Figure 3 essentially shows that the difference of accuracy between FCI and MBCS* is not significant in either way. On each learning problem, the returned PAGs have been checked for correctness with respect to the maximally oriented PAG $\mathcal{G}_0$ theoretically obtainable (as returned by the first series of experiments). The discrepancies were classified either as edge errors (when an arc was missing or extraneous in the returned PAG w.r.t. $\mathcal{G}_0$), or orientation errors (when a predicted arc in the returned PAG was indeed present in $\mathcal{G}_0$, but had a reversed direction or different end points). On all 8 learning problems, both edge and orientation errors are similar within the margin indicated by the standard-deviation error bars.

Note that the overall relatively high error rate comes from the failure of statistical tests with limited sample size. This indicates that structure learning is a hard problem and that low-sample-size situations where tests typically fail must be investigated further.

# 6 Conclusion

With the formalism of MAGs and PAGs, it is possible to learn an independence-equivalence class of projections of latent structures. We have shown an algorithm, MBCS*, which is much more efficient than the reference FCI algorithms on networks that are sufficiently sparse, making up to three orders of magnitude fewer conditional-independence tests to retrieve the same structure. We have experimental evidence that structural accuracy of MBCS* is as good as that of FCI. MBCS* is based on a first phase that identifies the Markov blanket of the underlying MAG, and then makes local adjustments to remove the spurious links and identify all colliders. The last step involving orientation rules is the same as for FCI. The reduced practical complexity makes MBCS* solve in minutes problems that FCI would need several days to solve. In that sense, MBCS* makes a whole new range of problems computationally tractable.

**References**

X. Boyen, N. Friedman, and D. Koller. Discovering the hidden structure of complex dynamic systems. In *Proceedings of the 15th Conference on Uncertainty in Artificial Intelligence*, 1999.

G. Elidan and N. Friedman. Learning the dimensionality of hidden variables. In *Proceedings of the 17th Conference in Uncertainty in Artificial Intelligence*, pages 144–151, 2001.

G. Elidan, N. Lotner, N. Friedman, and D. Koller. Discovering hidden variables: A structure-based approach. In *Proceedings of the 13th Conference on Advances in Neural Information Processing Systems*, 2001.

D. Margaritis and S. Thrun. Bayesian network induction via local neighborhoods. In *Advances in Neural Information Processing Systems 12*, 1999.

J. Pearl. *Causality: Models, Reasoning, and Inference*. Cambridge University Press, 2000.

J. Pearl. *Probabilistic Reasoning in Intelligent Systems: Networks of Plausible Inference*. Morgan Kaufmann, Los Altos, 1988.

J. Pearl and T. Verma. A theory of inferred causation. In *Proc. of the Second Int. Conf. on Principles of Knowledge Representation and Reasoning*. Morgan Kaufmann, 1991.

J.-P. Pellet and A. Elisseeff. Using Markov blankets for causal structure learning. *Journal of Machine Learning Research*, 9:1295–1342, 2008.

R. Scheines. An introduction to causal inference. In V. McKim and S. Turner, editors, *Causality in Crisis?*, pages 185–200. Univ. of Notre Dame Press, 1997.

R. Scheines, P. Spirtes, C. Glymour, C. Meek, and T. Richardson. The TETRAD project: Constraint based aids to causal model specification. Technical report, Carnegie Mellon University, Dpt. of Philosophy, 1995.

R. Silva, R. Scheines, C. Glymour, and P. Spirtes. Learning the structure of linear latent variable models. *Journal of Machine Learning Research*, 7:191–246, 2006.

P. Spirtes, C. Meek, and T. Richardson. Causal inference in the presence of latent variables and selection bias. In Philippe Besnard and Steve Hanks, editors, *Proceedings of the 11th Conference on Uncertainty in Artificial Intelligence*, pages 491–498, San Mateo, CA, 1995. Morgan Kaufmann.

P. Spirtes, T. Richardson, and C. Meek. Heuristic greedy search algorithms for latent variable models. In *Proceedings of the 6th International Workshop on Artificial Intelligence and Statistics*, 1996.

P. Spirtes, C. Glymour, and R. Scheines. *Causation, Prediction, and Search, Second Edition*. The MIT Press, 2001. ISBN 0262194406.

T. Verma. Graphical aspects of causal models. Technical Report R-191, Cognitive Systems Laboratory, UCLA, 1993.

## Footnotes

[1]Available at `http://jp.pellet.name/publis/pellet08nips_supplement.pdf`.

[2]We would have liked to both vary the number of samples for each dataset and include more test networks, but the running times of FCI on the larger instances, even when run with an upper limit on the maximum size of conditioning sets, were prohibitive, ranging up to a week on dense networks on a 2 GHz machine.
